# A MEAN FIELD THEORY OF LAYER IV OF VISUAL CORTEX AND ITS APPLICATION TO ARTIFICIAL NEURAL NETWORKS*

Christopher L. Scofield
Center for Neural Science and Physics Department
Brown University
Providence, Rhode Island 02912
and
Nestor, Inc., 1 Richmond Square, Providence, Rhode Island, 02906.

## ABSTRACT

A single cell theory for the development of selectivity and ocular dominance in visual cortex has been presented previously by Bienenstock, Cooper and Munro[1]. This has been extended to a network applicable to layer IV of visual cortex[2]. In this paper we present a mean field approximation that captures in a fairly transparent manner the qualitative, and many of the quantitative, results of the network theory. Finally, we consider the application of this theory to artificial neural networks and show that a significant reduction in architectural complexity is possible.

## A SINGLE LAYER NETWORK AND THE MEAN FIELD APPROXIMATION

We consider a single layer network of ideal neurons which receive signals from outside of the layer and from cells within the layer (Figure 1). The activity of the $i^{th}$ cell in the network is

$$c_i = m_i \, d + \sum_j L_{ij} \, c_j. \tag{1}$$

$d$ is a vector of afferent signals to the network. Each cell receives input from n fibers outside of the cortical network through the matrix of synapses $m_i$. Intra-layer input to each cell is then transmitted through the matrix of cortico-cortical synapses L.

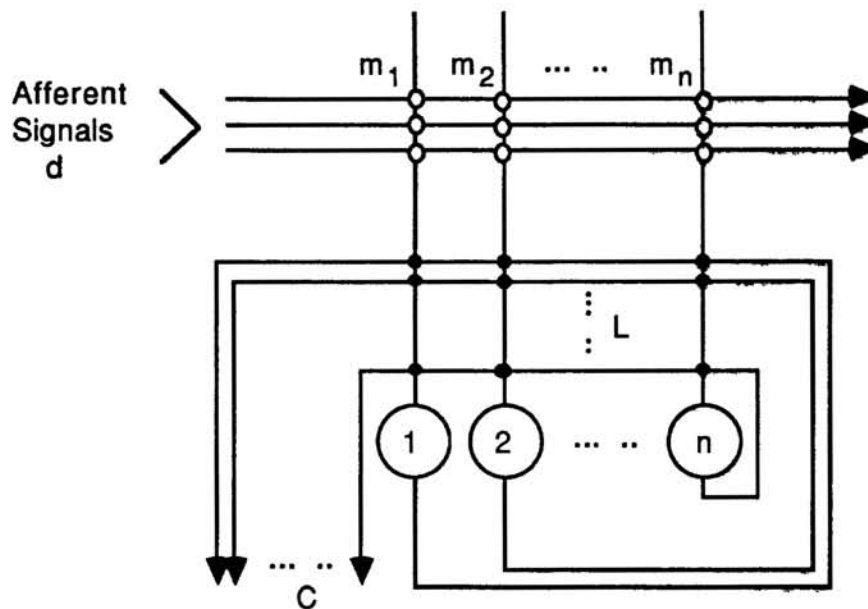

Figure 1: The general single layer recurrent network. Light circles are the LGN-cortical synapses. Dark circles are the (non-modifiable) cortico-cortical synapses.

We now expand the response of the $i^{th}$ cell into individual terms describing the number of cortical synapses traversed by the signal d before arriving through synapses $L_{ij}$ at cell i. Expanding $c_j$ in (1), the response of cell i becomes

$$c_i = m_i\, d + \sum_j L_{ij}\, m_j\, d + \sum_j L_{ij} \sum_k L_{jk}\, m_k\, d + \sum_j L_{ij} \sum_k L_{jk} \sum_n L_{kn}\, m_n\, d + ... \quad (2)$$

Note that each term contains a factor of the form

$$\sum_p L_{qp}\, m_p d.$$

This factor describes the first order effect, on cell q, of the cortical transformation of the signal d. The mean field approximation consists of estimating this factor to be a constant, independant of cell location

$$\sum_p L_{qp}\, m_p d = N\, \bar{m} d\, L_o = \text{constant}. \quad (3)$$

This assumption does not imply that each cell in the network is selective to the same pattern, (and thus that $m_i = m_j$). Rather, the assumption is that the vector sum is a constant

$$( \sum_p L_{qp} m_p) d = (N \bar{m} L_o) d.$$

This amounts to assuming that each cell in the network is surrounded by a population of cells which represent, on average, all possible pattern preferences. Thus the vector sum of the afferent synaptic states describing these pattern preferences is a constant independent of location.

Finally, if we assume that the lateral connection strengths are a function only of i-j then $L_{ij}$ becomes a circular matrix so that

$$\sum_i L_{ij} = \sum_j L_{ji} = L_o = \text{constant}.$$

Then the response of the cell i becomes

$$c_i = m_i d + ( L_o + L_o^2 + \ldots ) \bar{m} d. \qquad (4)$$

$$= m_i d + (N L_o /(1 - L_o )) \bar{c}, \qquad \text{for } | L_o | < 1$$

where we define the spatial average of cortical cell activity $\bar{c} = \bar{m} d$, and N is the average number of intracortical synapses.

Here, in a manner similar to that in the theory of magnetism, we have replaced the effect of individual cortical cells by their average effect (as though all other cortical cells can be replaced by an 'effective' cell, figure 2). Note that we have retained all orders of synaptic traversal of the signal d.

Thus, we now focus on the activity of the layer after 'relaxation' to equilibrium. In the mean field approximation we can therefore write

$$= (m_i - \alpha) d \qquad (5)$$

where the mean field

$$\alpha = a \bar{m}$$

with

$$a = N |L_o| (1 + |L_o|)^{-1},$$

and we asume that $L_0 < 0$ (the network is, on average, inhibitory).

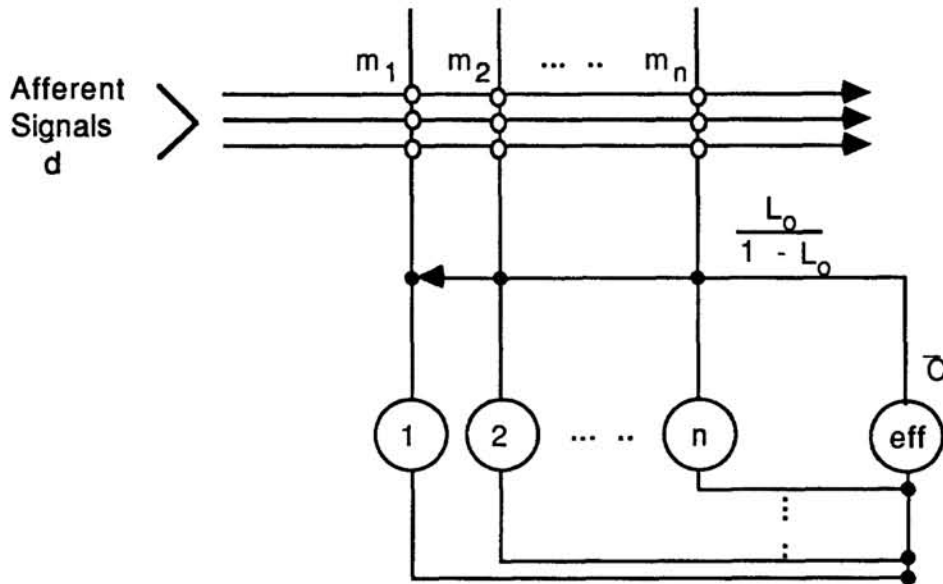

Figure 2: The single layer mean field network. Detailed connectivity between all cells of the network is replaced with a single (non-modifiable) synapse from an 'effective' cell.

## LEARNING IN THE CORTICAL NETWORK

We will first consider evolution of the network according to a synaptic modification rule that has been studied in detail, for single cells, elsewhere[1, 3]. We consider the LGN - cortical synapses to be the site of plasticity and assume for maximum simplicity that there is no modification of cortico-cortical synapses. Then

$$\dot{m}_i = \phi(c_i, \bar{\bar{c}}_i)\, d \tag{6}$$

$$\dot{L}_{ij} = 0.$$

In what follows $\bar{c}$ denotes the spatial average over cortical cells, while $\bar{\bar{c}}_i$ denotes the time averaged activity of the $i^{th}$ cortical cell. The function $\phi$ has been discussed extensively elsewhere. Here we note that $\phi$ describes a function of the cell response that has both hebbian and anti-hebbian regions.

This leads to a very complex set of non-linear stochastic equations that have been analyzed partially elsewhere[2]. In general, the afferent synaptic state has fixed points that are stable and selective and unstable fixed points that are non-selective[1,2]. These arguments may now be generalized for the network. In the mean field approximation

$$\dot{m}_i(\alpha) = \phi(c_i(\alpha), \bar{\bar{c}}_i(\alpha)) \, d = \phi[m_i(\alpha) - \alpha] \, d \qquad (7)$$

The mean field, $\alpha$ has a time dependent component $\bar{m}$. This varies as the average over all of the network modifiable synapses and, in most environmental situations, should change slowly compared to the change of the modifiable synapses to a single cell. Then in this approximation we can write

$$\overset{\bullet}{(m_i(\alpha) - \alpha)} = \phi[m_i(\alpha) - \alpha] \, d. \qquad (8)$$

We see that there is a mapping

$$m_i' <\longrightarrow m_i(\alpha) - \alpha \qquad (9)$$

such that for every $m_i(\alpha)$ there exists a corresponding (mapped) point $m_i'$ which satisfies

$$\dot{m}_i' = \phi[m_i'] \, d,$$

the original equation for the mean field zero theory. It can be shown [2,4] that for every fixed point of $m_i(\alpha = 0)$, there exists a corresponding fixed point $m_i(\alpha)$ with the same selectivity and stability properties. The fixed points are available to the neurons if there is sufficient inhibition in the network ($|L_o|$ is sufficiently large).

## APPLICATION OF THE MEAN FIELD NETWORK TO LAYER IV OF VISUAL CORTEX

Neurons in the primary visual cortex of normal adult cats are sharply tuned for the orientation of an elongated slit of light and most are activated by stimulation of either eye. Both of these properties--orientation selectivity and binocularity--depend on the type of visual environment experienced during a critical

period of early postnatal development. For example, deprivation of patterned input during this critical period leads to loss of orientation selectivity while monocular deprivation (MD) results in a dramatic shift in the ocular dominance of cortical neurons such that most will be responsive exclusively to the open eye. The ocular dominance shift after MD is the best known and most intensively studied type of visual cortical plasticity.

The behavior of visual cortical cells in various rearing conditions suggests that some cells respond more rapidly to environmental changes than others. In monocular deprivation, for example, some cells remain responsive to the closed eye in spite of the very large shift of most cells to the open eye. Singer et. al.[5] found, using intracellular recording, that geniculo-cortical synapses on inhibitory interneurons are more resistant to monocular deprivation than are synapses on pyramidal cell dendrites. Recent work suggests that the density of inhibitory GABAergic synapses in kitten striate cortex is also unaffected by MD during the cortical period [6, 7].

These results suggest that some LGN-cortical synapses modify rapidly, while others modify relatively slowly, with slow modification of some cortico-cortical synapses. Excitatory LGN-cortical synapses into excitatory cells may be those that modify primarily. To embody these facts we introduce two types of LGN-cortical synapses: those ($m_i$) that modify and those ($z_k$) that remain relatively constant. In a simple limit we have

$$\dot{m}_i = \phi(c_i, \bar{\bar{c}}_i)\, d$$

and                                                                                     (10)

$$\dot{z}_k = 0.$$

We assume for simplicity and consistent with the above physiological interpretation that these two types of synapses are confined to two different classes of cells and that both left and right eye have similar synapses (both $m_i$ or both $z_k$) on a given cell. Then, for binocular cells, in the mean field approximation (where binocular terms are in italics)

$$c_i(\alpha) = (m_i - \alpha)d = (m_i^l - \alpha^l)\cdot d^l + (m_i^r - \alpha^r)\cdot d^r$$

$$c_k(\alpha) = (z_k - \alpha)d = (z_k^l - \alpha^l)\cdot d^l + (z_k^r - \alpha^r)\cdot d^r,$$

where $d^{l(r)}$ are the explicit left (right) eye time averaged signals arriving form the LGN. Note that $\alpha^{l(r)}$ contain terms from modifiable and non-modifiable synapses:

$$\alpha^{l(r)} = a\,(\bar{m}^{l(r)} + \bar{z}^{l(r)}).$$

Under conditions of monocular deprivation, the animal is reared with one eye closed. For the sake of analysis assume that the right eye is closed and that only noise-like signals arrive at cortex from the right eye. Then the environment of the cortical cells is:

$$d = (d^j, n) \tag{12}$$

Further, assume that the left eye synapses have reached their selective fixed point, selective to pattern $d^1$. Then $(m_i^l, m_i^r) =$

$(m_i^{l*}, x_i)$ with $|x_i| \ll |m_i^{l*}|$. Following the methods of BCM, a local

linear analysis of the $\phi$ - function is employed to show that for the closed eye

$$x_i = a\,(1 - \lambda a)^{-1}\bar{z}^r. \tag{13}$$

where $\lambda = N_m/N$ is the ratio of the number modifiable cells to the total number of cells in the network. That is, the asymptotic state of the closed eye synapses is a scaled function of the mean-field due to non-modifiable (inhibitory) cortical cells. The scale of this state is set not only by the proportion of non-modifiable cells, but in addition, by the averaged intracortical synaptic strength $L_o$.

Thus contrasted with the mean field zero theory the deprived eye LGN-cortical synapses do not go to zero. Rather they approach the constant value dependent on the average inhibition produced by the non-modifiable cells in such a way that the asymptotic output of the cortical cell is zero (it cannot be driven by the deprived eye). However lessening the effect of inhibitory synapses (e.g. by application of an inhibitory blocking agent such as bicuculine) reduces the magnitude of $\alpha$ so that one could once more obtain a response from the deprived eye.

We find, consistent with previous theory and experiment, that most learning can occur in the LGN-cortical synapse, for inhibitory (cortico-cortical) synapses need not modify. Some non-modifiable LGN-cortical synapses are required.

## THE MEAN FIELD APPROXIMATION AND ARTIFICIAL NEURAL NETWORKS

The mean field approximation may be applied to networks in which the cortico-cortical feedback is a general function of cell activity. In particular, the feedback may measure the difference between the network activity and memories of network activity. In this way, a network may be used as a content addressable memory. We have been discussing the properties of a mean field network after equilibrium has been reached. We now focus on the detailed time dependence of the relaxation of the cell activity to a state of equilibrium.

Hopfield[8] introduced a simple formalism for the analysis of the time dependence of network activity. In this model, network activity is mapped onto a physical system in which the state of neuron activity is considered as a 'particle' on a potential energy surface. Identification of the pattern occurs when the activity 'relaxes' to a nearby minima of the energy. Thus minima are employed as the sites of memories. For a Hopfield network of N neurons, the intra-layer connectivity required is of order $N^2$. This connectivity is a significant constraint on the practical implementation of such systems for large scale problems. Further, the Hopfield model allows a storage capacity which is limited to m < N memories[8, 9]. This is a result of the proliferation of unwanted local minima in the 'energy' surface.

Recently, Bachmann et al.[10], have proposed a model for the relaxation of network activity in which memories of activity patterns are the sites of negative 'charges', and the activity caused by a test pattern is a positive test 'charge'. Then in this model, the energy function is the electrostatic energy of the (unit) test charge with the collection of charges at the memory sites

$$E = -1/L \sum_j Q_j \, | \mu - x_j | - L, \qquad (14)$$

where $\mu(0)$ is a vector describing the initial network activity caused by a test pattern, and $x_j$, the site of the $j^{th}$ memory. L is a parameter related to the network size.

This model has the advantage that storage density is not restricted by the the network size as it is in the Hopfield model, and in addition, the architecture employs a connectivity of order $m \times N$. Note that at each stage in the settling of $\mu(t)$ to a memory (of network activity) $x_j$, the only feedback from the network to each cell is the scalar

$$\sum_j Q_j \mid \mu - x_j \mid^{-L} \qquad (15)$$

This quantity is an integrated measure of the distance of the current network state from stored memories. Importantly, this measure is the same for all cells; it is as if a single virtual cell was computing the distance in activity space between the current state and stored states. The result of the computation is then broadcast to all of the cells in the network. This is a generalization of the idea that the detailed activity of each cell in the network need not be fed back to each cell. Rather some global measure, performed by a single 'effective' cell is all that is sufficient in the feedback.

DISCUSSION

We have been discussing a formalism for the analysis of networks of ideal neurons based on a mean field approximation of the detailed activity of the cells in the network. We find that a simple assumption concerning the spatial distribution of the pattern preferences of the cells allows a great simplification of the analysis. In particular, the detailed activity of the cells of the network may be replaced with a mean field that in effect is computed by a single 'effective' cell.

Further, the application of this formalism to the cortical layer IV of visual cortex allows the prediction that much of learning in cortex may be localized to the LGN-cortical synaptic states, and that cortico-cortical plasticity is relatively unimportant. We find, in agreement with experiment, that monocular deprivation of the cortical cells will drive closed-eye responses to zero, but chemical blockage of the cortical inhibitory pathways would reveal non-zero closed-eye synaptic states.

Finally, the mean field approximation allows the development of single layer models of memory storage that are unrestricted in storage density, but require a connectivity of order m×N. This is significant for the fabrication of practical content addressable memories.

## ACKNOWLEDGEMENTS

I would like to thank Leon Cooper for many helpful discussions and the contributions he made to this work.

*This work was supported  by the Office of Naval Research and the Army Research Office under contracts #N00014-86-K-0041 and #DAAG-29-84-K-0202.

## REFERENCES

[1] Bienenstock, E. L., Cooper, L. N & Munro, P. W. (1982) *J. Neuroscience* **2**, 32-48.
[2] Scofield, C. L. (1984) Unpublished Dissertation.
[3] Cooper, L. N, Munro, P. W. & Scofield, C. L. (1985) in *Synaptic Modification, Neuron Selectivity and Nervous System Organization*, ed. C. Levy, J. A. Anderson & S. Lehmkuhle, (Erlbaum Assoc., N. J.).
[4] Cooper, L. N & Scofield, C. L. (to be published) *Proc. Natl. Acad. Sci. USA*..
[5] Singer, W. (1977) **Brain Res. 134**, 508-000.
[6] Bear, M. F., Schmechel D. M., & Ebner, F. F. (1985) *J. Neurosci.* **5**, 1262-0000.
[7] Mower, G. D., White, W. F., & Rustad, R. (1986) *Brain Res.* **380**, 253-000.
[8] Hopfield, J. J. (1982) *Proc. Natl. Acad.* Sci. USA **79**, 2554-2558.
[9] Hopfield, J. J., Feinstein, D. I., & Palmer, R. G. (1983) *Nature* **304**, 158-159.
[10] Bachmann, C. M., Cooper, L. N, Dembo, A. & Zeitouni, O. (to be published) *Proc. Natl. Acad. Sci. USA.*